# Generalized Learning Vector Quantization

**Atsushi Sato & Keiji Yamada**
Information Technology Research Laboratories,
NEC Corporation
1-1, Miyazaki 4-chome, Miyamae-ku,
Kawasaki, Kanagawa 216, Japan
E-mail: {asato, yamada}@pat.cl.nec.co.jp

## Abstract

We propose a new learning method, "Generalized Learning Vector Quantization (GLVQ)," in which reference vectors are updated based on the steepest descent method in order to minimize the cost function. The cost function is determined so that the obtained learning rule satisfies the convergence condition. We prove that Kohonen's rule as used in LVQ does not satisfy the convergence condition and thus degrades recognition ability. Experimental results for printed Chinese character recognition reveal that GLVQ is superior to LVQ in recognition ability.

## 1  INTRODUCTION

Artificial neural network models have been applied to character recognition with good results for small-set characters such as alphanumerics (Le Cun *et al.*, 1989) (Yamada *et al.*, 1989). However, applying the models to large-set characters such as Japanese or Chinese characters is difficult because most of the models are based on Multi-Layer Perceptron (MLP) with the back propagation algorithm, which has a problem in regard to local minima as well as requiring a lot of calculation.

Classification methods based on pattern matching have commonly been used for large-set character recognition. Learning Vector Quantization (LVQ) has been studied to generate optimal reference vectors because of its simple and fast learning algorithm (Kohonen, 1989; 1995). However, one problem with LVQ is that reference vectors diverge and thus degrade recognition ability. Much work has been done on improving LVQ (Lee & Song, 1993) (Miyahara & Yoda, 1993) (Sato & Tsukumo, 1994), but the problem remains unsolved.

Recently, a generalization of the Simple Competitive Learning (SCL) has been under

study (Pal *et al.*, 1993) (Gonzalez *et al.*, 1995), and one unsupervised learning rule has been derived based on the steepest descent method to minimize the cost function. Pal *et al.* call their model "Generalized Learning Vector Quantization," but it is not a generalization of Kohonen's LVQ.

In this paper, we propose a new learning method for supervised learning, in which reference vectors are updated based on the steepest descent method, to minimize the cost function. This is a generalization of Kohonen's LVQ, so we call it "Generalized Learning Vector Quantization (GLVQ)." The cost function is determined so that the obtained learning rule satisfies the convergence condition. We prove that Kohonen's rule as used in LVQ does not satisfy the convergence condition and thus degrades recognition ability. Preliminary experiments revealed that non-linearity in the cost function is very effective for improving recognition ability. Printed Chinese character recognition experiments were carried out, and we can show that the recognition ability of GLVQ is very high compared with LVQ.

## 2   REVIEW OF LVQ

Assume that a number of reference vectors $w_k$ are placed in the input space. Usually, several reference vectors are assigned to each class. An input vector $x$ is decided to belong to the same class to which the nearest reference vector belongs. Let $w_k(t)$ represent sequences of the $w_k$ in the discrete-time domain. Heretofore, several LVQ algorithms have been proposed (Kohonen, 1995), but in this section, we will focus on LVQ2.1. Starting with properly defined initial values, the reference vectors are updated as follows by the LVQ2.1 algorithm:

$$w_i(t+1) = w_i(t) - \alpha(t)(x - w_i(t)), \qquad (1)$$
$$w_j(t+1) = w_j(t) + \alpha(t)(x - w_j(t)), \qquad (2)$$

where $0 < \alpha(t) < 1$, and $\alpha(t)$ may decrease monotonically with time. The two reference vectors $w_i$ and $w_j$ are the nearest to $x$; $x$ and $w_j$ belong to the same class, while $x$ and $w_i$ belong to different classes. Furthermore, $x$ must fall into the "window," which is defined around the midplane of $w_i$ and $w_j$. That is, if the following condition is satisfied, $w_i$ and $w_j$ are updated:

$$\min\left(\frac{d_i}{d_j}, \frac{d_j}{d_i}\right) > s, \qquad (3)$$

where $d_i = |x - w_i|$, $d_j = |x - w_j|$. The LVQ2.1 algorithm is based on the idea of shifting the decision boundaries toward the Bayes limits with attractive and repulsive forces from $x$. However, no attention is given to what might happen to the location of the $w_k$, so the reference vectors diverge in the long run. LVQ3 has been proposed to ensure that the reference vectors continue approximating the class distributions, but it must be noted that if only one reference vector is assigned to each class, LVQ3 is the same as LVQ2.1, and the problem of reference vector divergence remains unsolved.

## 3   GENERALIZED LVQ

To ensure that the reference vectors continue approximating the class distributions, we propose a new learning method based on minimizing the cost function. Let $w_1$ be the nearest reference vector that belongs to the same class of $x$, and likewise let $w_2$ be the nearest reference vector that belongs to a different class from $x$. Let us consider the relative distance difference $\mu(x)$ defined as follows:

$$\mu(x) = \frac{d_1 - d_2}{d_1 + d_2}, \qquad (4)$$

where $d_1$ and $d_2$ are the distances of $\boldsymbol{x}$ from $\boldsymbol{w}_1$ and $\boldsymbol{w}_2$, respectively. $\mu(\boldsymbol{x})$ ranges between $-1$ and $+1$, and if $\mu(\boldsymbol{x})$ is negative, $\boldsymbol{x}$ is classified correctly; otherwise, $\boldsymbol{x}$ is classified incorrectly. In order to improve error rates, $\mu(\boldsymbol{x})$ should decrease for all input vectors. Thus, a criterion for learning is formulated as the minimizing of a cost function $S$ defined by

$$S = \sum_{i=1}^{N} f(\mu(\boldsymbol{x}_i)), \tag{5}$$

where $N$ is the number of input vectors for training, and $f(\mu)$ is a monotonically increasing function. To minimize $S$, $\boldsymbol{w}_1$ and $\boldsymbol{w}_2$ are updated based on the steepest descent method with a small positive constant $\alpha$ as follows:

$$\boldsymbol{w}_i \leftarrow \boldsymbol{w}_i - \alpha \frac{\partial S}{\partial \boldsymbol{w}_i}, \quad i = 1, 2 \tag{6}$$

If squared Euclid distance, $d_i = |\boldsymbol{x} - \boldsymbol{w}_i|^2$, is used, we can obtain the following.

$$\frac{\partial S}{\partial \boldsymbol{w}_1} = \frac{\partial S}{\partial \mu} \frac{\partial \mu}{\partial d_1} \frac{\partial d_1}{\partial \boldsymbol{w}_1} = -\frac{\partial f}{\partial \mu} \frac{4 d_2}{(d_1 + d_2)^2} (\boldsymbol{x} - \boldsymbol{w}_1) \tag{7}$$

$$\frac{\partial S}{\partial \boldsymbol{w}_2} = \frac{\partial S}{\partial \mu} \frac{\partial \mu}{\partial d_2} \frac{\partial d_2}{\partial \boldsymbol{w}_2} = +\frac{\partial f}{\partial \mu} \frac{4 d_1}{(d_1 + d_2)^2} (\boldsymbol{x} - \boldsymbol{w}_2) \tag{8}$$

Therefore, the GLVQ's learning rule can be described as follows:

$$\boldsymbol{w}_1 \leftarrow \boldsymbol{w}_1 + \alpha \frac{\partial f}{\partial \mu} \frac{d_2}{(d_1 + d_2)^2} (\boldsymbol{x} - \boldsymbol{w}_1) \tag{9}$$

$$\boldsymbol{w}_2 \leftarrow \boldsymbol{w}_2 - \alpha \frac{\partial f}{\partial \mu} \frac{d_1}{(d_1 + d_2)^2} (\boldsymbol{x} - \boldsymbol{w}_2) \tag{10}$$

Let us discuss the meaning of $f(\mu)$. $\partial f / \partial \mu$ is a kind of gain factor for updating, and its value depends on $\boldsymbol{x}$. In other words, $\partial f / \partial \mu$ is a weight for each $\boldsymbol{x}$. To decrease the error rate, it is effective to update reference vectors mainly by input vectors around class boundaries, so that the decision boundaries are shifted toward the Bayes limits. Accordingly, $f(\mu)$ should be a *non-linear* monotonically increasing function, and it is considered that classification ability depends on the definition of $f(\mu)$. In this paper, $\partial f / \partial \mu = f(\mu, t)\{1 - f(\mu, t)\}$ was used in the experiments, where $t$ is learning time and $f(\mu, t)$ is a sigmoid function of $1/(1 + e^{-\mu t})$. In this case, $\partial f / \partial \mu$ has a single peak at $\mu = 0$, and the peak width becomes narrower as $t$ increases, so the input vectors that affect learning are gradually restricted to those around the decision boundaries.

Let us discuss the meaning of $\mu$. $\boldsymbol{w}_1$ and $\boldsymbol{w}_2$ are updated by attractive and repulsive forces from $\boldsymbol{x}$, respectively, as shown in Eqs. (9) and (10), and the quantities of updating, $|\Delta \boldsymbol{w}_1|$ and $|\Delta \boldsymbol{w}_2|$, depend on derivatives of $\mu$. Reference vectors will converge to the equilibrium states defined by attractive and repulsive forces, so it is considered that convergence property depends on the definition of $\mu$.

## 4   DISCUSSION

First, we show that the conventional LVQ algorithms can be derived based on the framework of GLVQ. If $\mu = d_1$ for $d_1 < d_2$, $\mu = -d_2$ for $d_1 > d_2$, and $f(\mu) = \mu$, the cost function is written as $S = \sum_{d_1 < d_2} d_1 - \sum_{d_1 > d_2} d_2$. Then, we can obtain the following:

$$\boldsymbol{w}_1 \leftarrow \boldsymbol{w}_1 + \alpha(\boldsymbol{x} - \boldsymbol{w}_1), \ \boldsymbol{w}_2 \leftarrow \boldsymbol{w}_2 \qquad \text{for } d_1 < d_2 \tag{11}$$

$$\boldsymbol{w}_2 \leftarrow \boldsymbol{w}_2 - \alpha(\boldsymbol{x} - \boldsymbol{w}_2), \ \boldsymbol{w}_1 \leftarrow \boldsymbol{w}_1 \qquad \text{for } d_1 > d_2 \tag{12}$$

This learning algorithm is the same as LVQ1. If $\mu = d_1 - d_2$ and $f(\mu) = \mu$ for $|\mu| < s$, $f(\mu) = \text{const}$ for $|\mu| > s$, the cost function is written as $S = \sum_{|\mu| < s}(d_1 - d_2) + C$. Then, we can obtain the following:

if $|\mu| < s$ ($x$ falls into the window)

$$w_1 \leftarrow w_1 + \alpha(x - w_2) \tag{13}$$

$$w_2 \leftarrow w_2 - \alpha(x - w_2) \tag{14}$$

In this case, $w_1$ and $w_2$ are updated simultaneously, and this learning algorithm is the same as LVQ2.1. So it can be said that GLVQ is a generalized model that includes the conventional LVQs.

Next, we discuss the convergence condition. We can obtain other learning algorithms by defining a different cost function, but it must be noted that the convergence property depends on the definition of the cost function. The main difference between GLVQ and LVQ2.1 is the definition of $\mu$; $\mu = (d_1 - d_2)/(d_1 + d_2)$ in GLVQ, $\mu = d_1 - d_2$ in LVQ2.1. Why do the reference vectors diverge in LVQ2.1, while they converge in GLVQ, as shown later? In order to clarify the convergence condition, let us consider the following learning rule:

$$w_1 \leftarrow w_1 + \alpha|x - w_2|^k(x - w_1) \tag{15}$$

$$w_2 \leftarrow w_2 - \alpha|x - w_1|^k(x - w_2) \tag{16}$$

Here, $|\Delta w_1|$ and $|\Delta w_2|$ are the quantities of updating by the attractive and the repulsive forces, respectively. The ratio of these two is calculated as follows:

$$\frac{|\Delta w_1|}{|\Delta w_2|} = \frac{\alpha|x - w_2|^k|x - w_1|}{\alpha|x - w_1|^k|x - w_2|} = \frac{|x - w_2|^{k-1}}{|x - w_1|^{k-1}} \tag{17}$$

If the initial values of reference vectors are properly defined, most $x$'s will satisfy $|x - w_1| < |x - w_2|$. Therefore, if $k > 1$, the attractive force is greater than the repulsive force, and the reference vectors will converge, because the attractive forces come from $x$'s that belong to the same class of $w_1$. In GLVQ, $k = 2$ as shown in Eqs. (9) and (10), and the vectors will converge, while they will diverge in LVQ2.1 because $k = 0$. According to the above discussion, we can use $d_i/(d_1 + d_2)$ or just $d_i$, instead of $d_i/(d_1 + d_2)^2$ in Eqs. (9) and (10). This correction does not affect the convergence condition. The essential problem in LVQ2.1 results from the drawback in Kohonen's rule with $k = 0$. In other words, the cost function used in LVQ is not determined so that the obtained learning rule satisfies the convergence condition.

## 5   EXPERIMENTS

### 5.1   PRELIMINARY EXPERIMENTS

The experimental results using Eqs. (15) and (16) with $\alpha = 0.001$, shown in Fig. 1, support the above discussion on the convergence condition. Two-dimensional input vectors with two classes shown in Fig. 1(a) were used in the experiments. The ideal decision boundary that minimizes the error rate is shown by the broken line. One reference vector was assigned to each class with initial values $(x, y) = (0.3, 0.5)$ for Class A and $(x, y) = (0.7, 0.5)$ for Class B. Figure 1(b) shows the distance between the two reference vectors during learning. The distance remains the same value for $k > 1$, while it increases with time for $k \leq 1$; that is, the reference vectors diverge.

Figure 2 shows the experimental results from GLVQ for linearly non-separable patterns compared with LVQ2.1. The input vectors shown in Fig. 2(a) were obtained by shifting all input vectors shown in Fig. 1(a) to the right by $|y - 0.5|$. The ideal

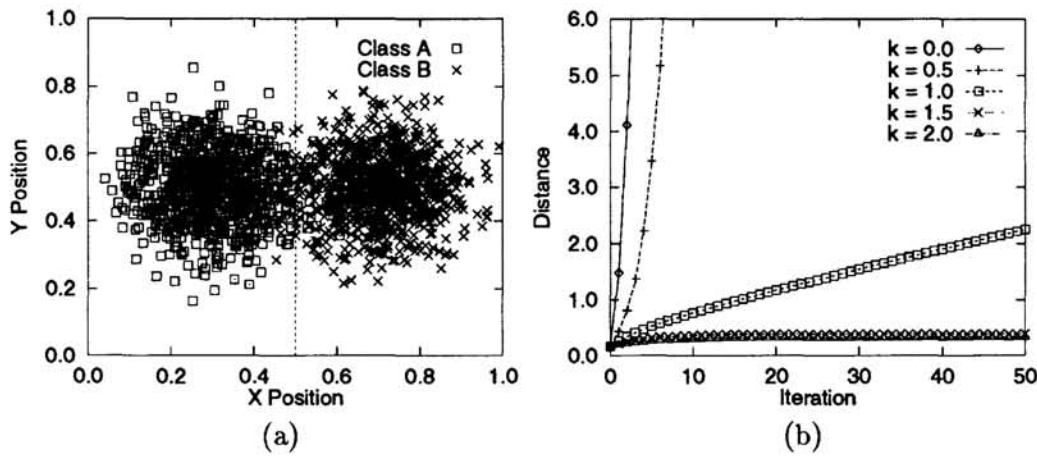

Figure 1: Experimental results that support the discussion on the convergence condition with one reference vector for each class. (a) Input vectors used in the experiments. The broken line shows the ideal decision boundary. (b) Distance between two reference vectors for each $k$ value during learning. The distance remains the same value for $k > 1$, while it diverges for $k \leq 1$.

decision boundary that minimizes the error rate is shown by the broken line. Two reference vectors were assigned to each class with initial values $(x, y) = (0.3, 0.4)$ and $(0.3, 0.6)$ for Class A, and $(x, y) = (0.7, 0.4)$ and $(0.7, 0.6)$ for Class B. The gain factor $\alpha$ was 0.004 in GLVQ and LVQ2.1, and the window parameter $s$ in LVQ2.1 was 0.8 in the experiments.

Figure 2(b) shows the number of error counts for all the input vectors during learning. GLVQ(NL) shows results by GLVQ with a non-linear function; that is, $\partial f / \partial \mu = f(\mu, t)\{1 - f(\mu, t)\}$. The number of error counts decreased with time to the minimum determined by the Bayes limit. GLVQ(L) shows results by GLVQ with a linear function; that is, $\partial f / \partial \mu = 1$. The number of error counts did not decrease to the minimum. This indicates that non-linearity of the cost function is very effective for improving recognition ability. Results using LVQ2.1 show that the number of error counts decreased in the beginning, but overall increased gradually with time. The degradation in the recognition ability results from the divergence of the reference vectors, as we have mentioned earlier.

## 5.2   CHARACTER RECOGNITION EXPERIMENTS

Printed Chinese character recognition experiments were carried out to examine the performance of GLVQ. Thirteen kinds of printed fonts with 500 classes were used in the experiments. The total number of characters was 13,000; half of which were used as training data, and the other half were used as test data. As input vectors, 256-dimensional orientation features were used (Hamanaka *et al.*, 1993). Only one reference vector was assigned to each class, and their initial values were defined by averaging training data for each class.

Recognition results for test data are tabulated in Table 1 compared with other methods. TM is the template matching method using mean vectors. LVQ2 is the earlier version of LVQ2.1. The learning algorithm is the same as LVQ2.1 described in Section 2, but $d_i$ must be less than $d_j$. The gain factor $\alpha$ was 0.05, and the window parameter $s$ was 0.65 in the experiments. The experimental result by LVQ3 was

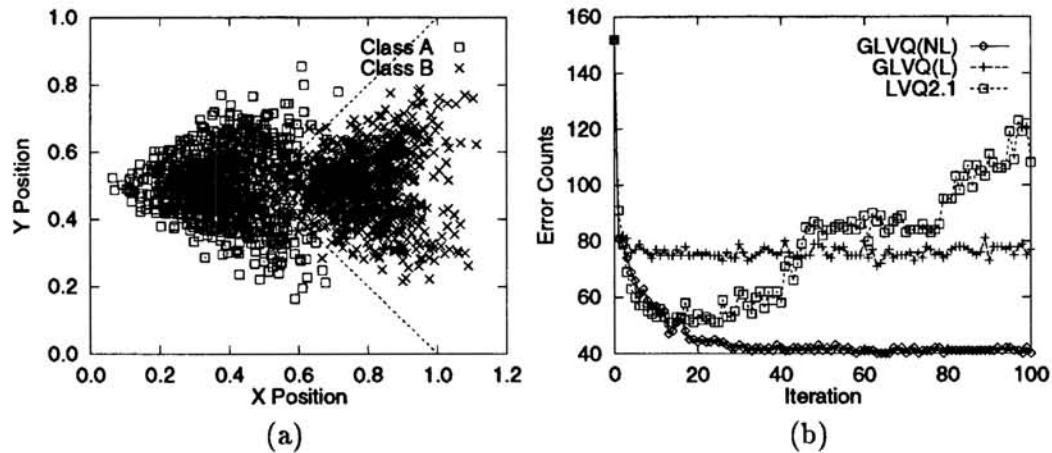

(a)                                                                    (b)

Figure 2: Experimental results for linearly non-separable patterns with two reference vectors for each class. (a) Input vectors used in the experiments. The broken line shows the ideal decision boundary. (b) The number of error counts during learning. GLVQ (NL) and GLVQ (L) denote the proposed method using a non-linear and linear function in the cost function, respectively. This shows that non-linearity of the cost function is very effective for improving classification ability.

Table 1: Experimental results for printed Chinese character recognition compared with other methods.

| Methods | Error rates(%) |
|---------|----------------|
| TM[1]   | 0.23           |
| LVQ2[2] | 0.18           |
| LVQ2.1  | 0.11           |
| IVQ[3]  | 0.08           |
| GLVQ    | 0.05           |

[1] Template matching using mean vectors.
[2] The earlier version of LVQ2.1.
[3] Our previous model (Improved Vector Quantization).

the same as that by LVQ2.1, because only one reference vector was assigned to each class. IVQ (Improved Vector Quantization) is our previous model based on Kohonen's rule (Sato & Tsukumo, 1994).

The error rate was extremely low for GLVQ, and a recognition rate of 99.95% was obtained. Ambiguous results can be rejected by thresholding the value of $\mu(x)$. If input vectors with $\mu(x) \geq -0.02$ were rejected, a recognition rate of 100% would be obtained, with a rejection rate of 0.08% for this experiment.

## 6   CONCLUSION

We proposed the Generalized Learning Vector Quantization as a new learning method. We formulated the criterion for learning as the minimizing of the cost function, and obtained the learning rule based on the steepest descent method. GLVQ is a generalized method that includes LVQ. We discussed the convergence condition and showed that the convergence property depends on the definition of

the cost function. We proved that the essential problem of the divergence of the reference vectors in LVQ2.1 results from a drawback of Kohonen's rule that does not satisfy the convergence condition. Preliminary experiments revealed that non-linearity in the cost function is very effective for improving recognition ability. We carried out printed Chinese character recognition experiments and obtained a recognition rate of 99.95%. The experimental results revealed that GLVQ is superior to the conventional LVQ algorithms.

## Acknowledgements

We are indebted to Mr. Jun Tsukumo and our colleagues in the Pattern Recognition Research Laboratory for their helpful cooperation.

## References

Y. Le Cun, B. Bose, J. S. Denker, D. Henderson, R. E. Howard, W. Hubbard, and L. D. Jackel, "Handwritten Digit Recognition with a Back-Propagation Network," *Neural Information Processing Systems 2*, pp. 396–404 (1989).

K. Yamada, H. Kami, J. Tsukumo, and T. Temma, "Handwritten Numeral Recognition by Multi-Layered Neural Network with Improved Learning Algorithm," *Proc. of the International Joint Conference on Neural Networks 89*, Vol. 2, pp. 259–266 (1989).

T. Kohonen, *Self-Organization and Associative Memory, 3rd ed.*, Springer-Verlag (1989).

T. Kohonen, "LVQ_PAK Version 3.1 — The Learning Vector Quantization Program Package," *LVQ Programming Team of the Helsinki University of Technology*, (1995).

S. W. Lee and H. H. Song, "Optimal Design of Reference Models Using Simulated Annealing Combined with an Improved LVQ3," *Proc. of the International Conference on Document Analysis and Recognition*, pp. 244–249 (1993).

K. Miyahara and F. Yoda, "Printed Japanese Character Recognition Based on Multiple Modified LVQ Neural Network," *Proc. of the International Conference on Document Analysis and Recognition*, pp. 250–253 (1993).

A. Sato and J. Tsukumo, "A Criterion for Training Reference Vectors and Improved Vector Quantization," *Proc. of the International Conference on Neural Networks*, Vol. 1, pp.161–166 (1994).

N. R. Pal, J. C. Bezdek, and E. C.-K. Tsao, "Generalized Clustering Networks and Kohonen's Self-organizing Scheme," *IEEE Trans. of Neural Networks*, Vol. 4, No. 4, pp. 549–557 (1993).

A. I. Gonzalez, M. Graña, and A. D'Anjou, "An Analysis of the GLVQ Algorithm," *IEEE Trans. of Neural Networks*, Vol. 6, No. 4, pp. 1012–1016 (1995).

M. Hamanaka, K. Yamada, and J. Tsukumo, "On-Line Japanese Character Recognition Experiments by an Off-Line Method Based on Normalization-Cooperated Feature Extraction," *Proc. of the International Conference on Document Analysis and Recognition*, pp. 204–207 (1993).
